# Energetically Optimal Action Potentials

**Martin Stemmler**
BCCN and LMU Munich
Grosshadernerstr. 2,
Planegg, 82125 Germany

**Biswa Sengupta, Simon Laughlin, Jeremy Niven**
Department of Zoology,
University of Cambridge,
Downing Street, Cambridge CB2 3EJ, UK

## Abstract

Most action potentials in the nervous system take on the form of strong, rapid, and brief voltage deflections known as spikes, in stark contrast to other action potentials, such as in the heart, that are characterized by broad voltage plateaus. We derive the shape of the neuronal action potential from first principles, by postulating that action potential generation is strongly constrained by the brain's need to minimize energy expenditure. For a given height of an action potential, the least energy is consumed when the underlying currents obey the bang-bang principle: the currents giving rise to the spike should be intense, yet short-lived, yielding spikes with sharp onsets and offsets. Energy optimality predicts features in the biophysics that are not *per se* required for producing the characteristic neuronal action potential: sodium currents should be extraordinarily powerful and inactivate with voltage; both potassium and sodium currents should have kinetics that have a bell-shaped voltage-dependence; and the cooperative action of multiple 'gates' should start the flow of current.

## 1 The paradox

Nerve cells communicate with each other over long distances using spike-like action potentials, which are brief electrical events traveling rapidly down axons and dendrites. Each action potential is caused by an accelerating influx of sodium or calcium ions, depolarizing the cell membrane by forty millivolts or more, followed by repolarization of the cell membrane caused by an efflux of potassium ions. As different species of ions are swapped across the membrane during the action potential, ion pumps shuttle the excess ions back and restore the ionic concentration gradients.

If we label each ionic species by $\alpha$, the work $\Delta E$ done to restore the ionic concentration gradients is

$$\Delta E = RT\mathcal{V} \sum_{\alpha} \Delta[\alpha]_{\text{in}} \ln \frac{[\alpha]_{\text{out}}}{[\alpha]_{\text{in}}}, \qquad (1)$$

where $R$ is the gas constant, $T$ is the temperature in Kelvin, $\mathcal{V}$ is the cell volume, $[\alpha]_{\text{in}|\text{out}}$ is the concentration of ion $\alpha$ inside or outside the cell, and $\Delta[\alpha]_{\text{in}}$ is the concentration change inside the cell, which is assumed to be small relative to the total concentration. The sum $\sum_{\alpha} z_{\alpha}\Delta[\alpha] = 0$, where $z_{\alpha}$ is the charge on ion $\alpha$, as no net charge accumulates during the action potential and no net work is done by or on the electric field. Often, sodium ($\text{Na}^+$) and potassium ($\text{K}^+$) play the dominant role in generating action potentials, in which case $\Delta E = \Delta[\text{Na}]_{\text{in}}F\mathcal{V}(E_{\text{Na}} - E_{\text{K}})$, where $F$ is Faraday's constant, $E_{\text{Na}} = RT/F \ln\left([\text{Na}]_{\text{out}}/[\text{Na}]_{\text{in}}\right)$ is the reversal potential for $\text{Na}^+$, at which no net sodium current flows, and $E_{\text{K}} = RT/F \ln\left([\text{K}]_{\text{out}}/[\text{K}]_{\text{in}}\right)$. This estimate of the work done does not include heat (due to loss through the membrane resistance) or the work done by the ion channel proteins in changing their conformational state during the action potential.

Hence, the action potential's energetic cost to the cell is directly proportional to $\Delta[\text{Na}]_{\text{in}}$; taking into account that each $\text{Na}^+$ ion carries one elementary charge, the cost is also proportional to the

charge $Q_{\text{Na}}$ that accumulates inside the cell. A maximally efficient cell reduces the charge per spike to a minimum. If a cell fires action potentials at an average rate $f$, the cell's Na/K pumps must move $\text{Na}^+$ and $\text{K}^+$ ions in opposite directions, against their respective concentration gradients, to counteract an average inward $\text{Na}^+$ current of $f\,Q_{\text{Na}}$. Exhaustive measurements on myocytes in the heart, which expend tremendous amounts of energy to keep the heart beating, indicate that Na/K pumps expel $\sim 0.5\ \mu A/\text{cm}^2$ of $\text{Na}^+$ current at membrane potentials close to rest [1]. Most excitable cells, even when spiking, spend most of their time close to resting potential, and yet standard models for action potentials can easily lead to accumulating an ionic charge of up to 5 $\mu C/\text{cm}^2$ [2]; most of this accumulation occurs during a very brief time interval. If one were to take an isopotential nerve cell with the same density of ion pumps as in the heart, then such a cell would not be able to produce more than an action potential *once every ten seconds* on average. The brain should be effectively silent.

Clearly, this conflicts with what is known about the average firing rates of neurons in the brainstem or even the neocortex, which can sustain spiking up to at least 7 Hz [3]. Part of the discrepancy can be resolved by noting that nerve cells are not isopotential and that action potential generation occurs within a highly restricted area of the membrane. Even so, standard models of action potential generation waste extraordinary amounts of energy; recent evidence [4] points out that many mammalian cortical neurons are much more efficient.

As nature places a premium on energy consumption, we will argue that one can predict both the shape of the action potential and the underlying biophysics of the nonlinear, voltage-dependent ionic conductances from the principle of minimal energy consumption. After reviewing the ionic basis of action potentials, we first sketch how to compute the minimal energy cost for an *arbitrary* spike shape, and then solve for the optimal action potential shape with a given height. Finally, we show how minimal energy consumption explains *all* the dynamical features in the standard Hodgkin-Huxley (HH) model for neuronal dynamics that distinguish the brain's action potentials from other highly nonlinear oscillations in physics and chemistry.

## 2 Ionic basis of the action potential

In an excitable cell, synaptic drive forces the membrane permeability to different ions to change rapidly in time, producing the dynamics of the action potential. The current density $I_\alpha$ carried by an ion species $\alpha$ is given by the Goldman-Hodgkin-Katz (GHK) current equation[5, 6, 2], which assumes that ions are driven independently across the membrane under the influence of a constant electric field. $I_\alpha$ depends upon the ions membrane permeability, $P_\alpha$, its concentrations on either side of the membrane $[\alpha]_{\text{out}}$ and $[\alpha]_{\text{in}}$ and the voltage across the membrane, $V$, according to:

$$I_\alpha = P_\alpha \frac{z_\alpha^2 V F^2}{RT} \; \frac{[\alpha]_{\text{out}} - [\alpha]_{\text{in}} \exp\left(z_\alpha V F / RT\right)}{1 - \exp(z_\alpha V F / RT)}, \tag{2}$$

To produce the fast currents that generate APs, a subset of the membranes ionic permeabilities $P_\alpha$ are gated by voltage. Changes in the permeability $P_\alpha$ are not instantaneous; the voltage-gated permeability is scaled mathematically by gating variables $m(t)$ and $h(t)$ with their own time dependence. After separating constant from time-dependent components in the permeability, the voltage-gated permeability obeys

$$P_\alpha(t) = m(t)^r h(t)^s \qquad \text{such that} \qquad 0 \le P_\alpha(t) \le \bar{P}_\alpha,$$

where $r$ and $s$ are positive, and $\bar{P}_\alpha$ is the peak permeability to ion $\alpha$ when all channels for ion $\alpha$ are open. Gating is also referred to as activation, and the associated nonlinear permeabilities are called active. There are also passive, voltage-insensitive permeabilities that maintain the resting potential and depolarise the membrane to trigger action potentials.

The simplest possible kinetics for the gating variables are first order, involving only a single derivative in time. The steady state of each gating variable at a given voltage is determined by a Boltzmann function, to which the gating variables evolve:

$$\tau_m \frac{dm}{dt} = \sqrt[r]{\bar{P}_\alpha}\, m_\infty(V) - m(t)$$

$$\text{and} \quad \tau_h \frac{dh}{dt} = h_\infty(V) - h(t),$$

with $m_\infty(V) = \{1 + \exp((V - V_m)/s_m)\}^{-1}$ the Boltzmann function described by the slope $s_m > 0$ and the midpoint $V_m$; similarly, $h_\infty(V) = \{1 + \exp((V - V_h)/s_h)\}^{-1}$, but with $s_h < 0$. Scaling $m_\infty(V)$ by the $r^{\text{th}}$ root of the peak permeability $\bar{P}_\alpha$ is a matter of mathematical convenience.

We will consider both voltage-independent and voltage-dependent time constants, either setting $\tau_j = \tau_{j,0}$ to be constant, where $j \in \{m(t), h(t)\}$, or imposing a bell-shaped voltage dependence $\tau_j(V) = \tau_{j,0} \operatorname{sech}[s_j(V - V_j)]$

The synaptic, leak, and voltage-dependent currents drive the rate of change in the voltage across the membrane

$$C\frac{dV}{dt} = I_{\text{syn}} + I_{\text{leak}} + \sum_\alpha I_\alpha,$$

where the synaptic permeability and leak permeability are held constant.

## 3 Resistive and capacitive components of the energy cost

By treating the action potential as the charging and discharging of the cell membrane capacitance, the action potentials measured at the mossy fibre synapse in rats [4] or in mouse thalamocortical neurons [7] were found to be highly energy-efficient: the nonlinear, active conductances inject only slightly more current than is needed to charge a capacitor to the peak voltage of the action potential. The implicit assumption made here is that one can neglect the passive loss of current through the membrane resistance, known as the leak. Any passive loss must be compensated by additional charge, making this loss the primary target of the selection pressure that has shaped the dynamics of action potentials. On the other hand, the membrane capacitance at the site of AP initiation is generally modelled and experimentally confirmed [8] as being fairly constant around $1\ \mu F/cm^2$; in contrast, the propagation, but not generation, of AP's can be assisted by a reduction in the capacitance achieved by the myelin sheath that wraps some axons. As myelin would block the flow of ions, we posit that the specific capacitance cannot yield to selection pressure to minimise the work $W = Q_{\text{Na}}(E_{\text{Na}} - E_{\text{K}})$ needed for AP generation.

To address how the shape and dynamics of action potentials might have evolved to consume less energy, we first fix the action potential's shape and solve for the minimum charge $Q_{\text{Na}}$ *ab initio*, without treating the cell membrane as a pure capacitor. Regardless of the action potential's particular time-course $V(t)$, voltage-dependent ionic conductances must transfer $\text{Na}^+$ and $\text{K}^+$ charge to elicit an action potential. Figure 1 shows a generic action potential and the associated ionic currents, comparing the latter to the minimal currents required. The passive equivalent circuit for the neuron consists of a resistor in parallel with a capacitor, driven by a synaptic current. To charge the membrane to the peak voltage, a neuron in a high-conductance state [9, 10] may well lose more charge through the resistor than is stored on the capacitor. For neurons in a low-conductance state and for rapid voltage deflections from the resting potential, membrane capacitance will be the primary determinant of the charge.

## 4 The norm of spikes

How close can voltage-gated channels with realistic properties come to the minimal currents? What time-course for the action potential leads to the smallest minimal currents?

To answer these questions, we must solve a constrained optimization problem on the solutions to the nonlinear differential equations for the neuronal dynamics. To separate action potentials from mere small-amplitude oscillations in the voltage, we need to introduce a metric. Smaller action potentials consume less energy, provided the underlying currents are optimal, yet signalling between neurons depends on the action potential's voltage deflection reaching a minimum amplitude. Given the importance of the action potential's amplitude, we define an $L^p$ norm on the voltage wave-form $V(t)$ to emphasize the maximal voltage deflection:

$$\|V(t) - \langle V \rangle\|_p = \left\{ \int_0^T \|V(t) - \langle V \rangle\|^p \, dt \right\}^{\frac{1}{p}},$$

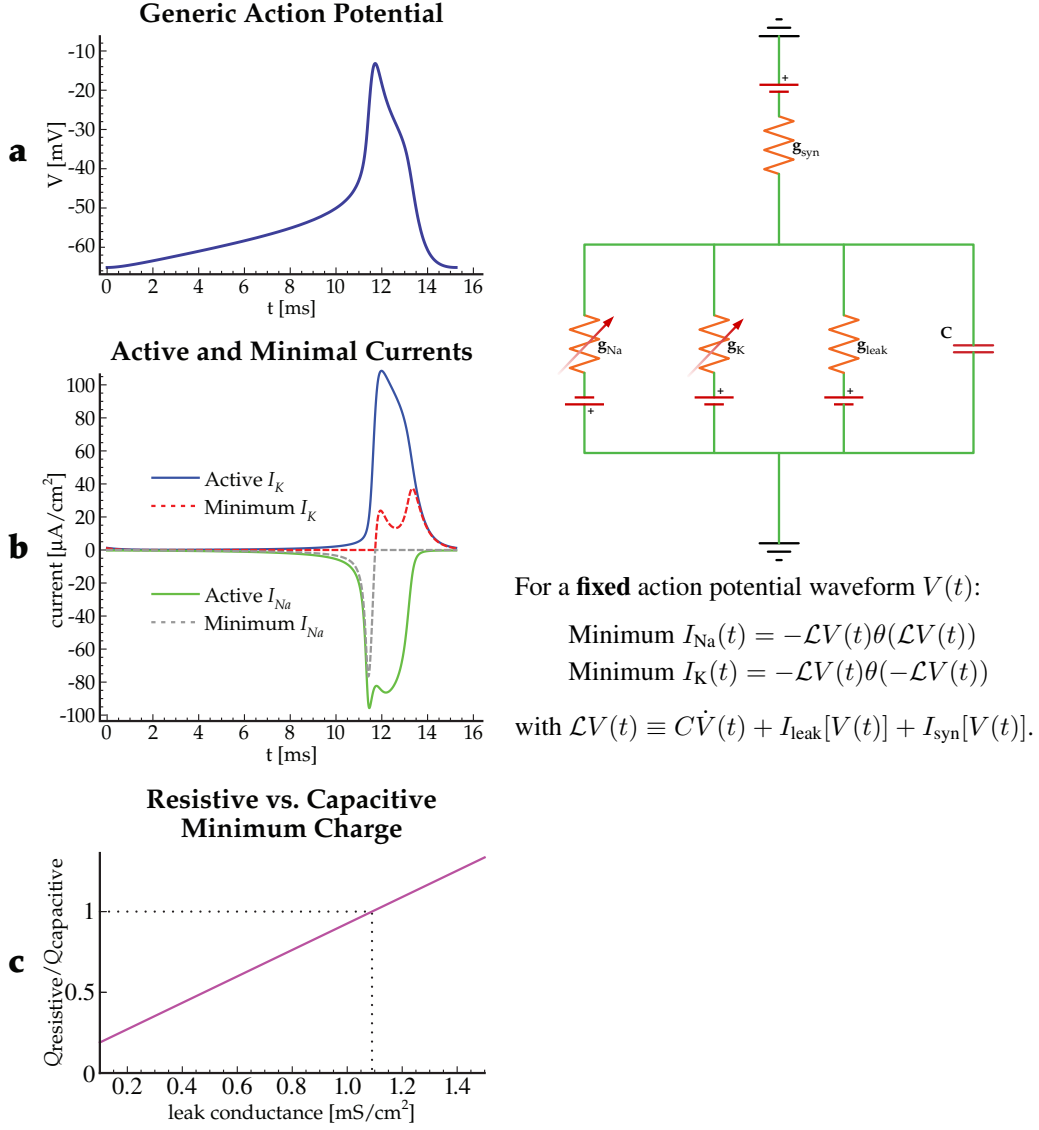

**Generic Action Potential**

**Active and Minimal Currents**

- Active $I_K$
- Minimum $I_K$
- Active $I_{Na}$
- Minimum $I_{Na}$

**Resistive vs. Capacitive Minimum Charge**

For a **fixed** action potential waveform $V(t)$:

$$\text{Minimum } I_{\text{Na}}(t) = -\mathcal{L}V(t)\theta(\mathcal{L}V(t))$$
$$\text{Minimum } I_{\text{K}}(t) = -\mathcal{L}V(t)\theta(-\mathcal{L}V(t))$$

with $\mathcal{L}V(t) \equiv C\dot{V}(t) + I_{\text{leak}}[V(t)] + I_{\text{syn}}[V(t)]$.

Figure 1: To generate an action potential with an arbitrary time-course $V(t)$, the nonlinear, time-dependent permeabilities must deliver more charge than just to load the membrane capacitance—resistive losses must be compensated. **(a)** The action potential's time-course in a generic HH model for a neuron, represented by the circuit diagram on the right. The peak of the action potential is $\sim 50$ mV above the average potential. **(b)** The inward Na$^+$ current, shown in green going in the negative direction, rapidly depolarizes the potential $V(t)$ and yields the upstroke of the action potential. Concurrently, the K$^+$ current activates, displayed as a positive deflection, and leads to the downstroke in the potential $V(t)$. Inward and outward currents overlap significantly in time. The dotted lines within the region bounded by the solid lines represent the minimal Na$^+$ current and the minimal K$^+$ current needed to produce the $V(t)$ spike waveform in (a). By the law of current conservation, the sum of capacitive, resistive, and synaptic currents, denoted by $\mathcal{L}V(t) \equiv C\dot{V}(t) + I_{\text{leak}}[V(t)] + I_{\text{syn}}[V(t)]$, must be balanced by the active currents. If the cell's passive properties, namely its capacitance and (leak) resistance, and the synaptic conductance are constant, we can deduce the *minimal* active currents needed to generate a specified $V(t)$. The minimal currents, by definition, do not overlap in time. Taking into account passive current flow, restoring the concentration gradients after the action potential requires 29 nJ/cm$^2$. By contrast, if the active currents were optimal, the cost would be 8.9 nJ/cm$^2$. **(c)** To depolarize from the minimum to the maximum of the AP, the synaptic voltage-gated currents must deliver a charge $Q_{\text{capacitive}}$ to charge the membrane capacitance and a charge $Q_{\text{resistive}}$ to compensate for the loss of current through leak channels. For a large leak conductance in the cell membrane, $Q_{\text{resistive}}$ can be larger than $Q_{\text{capacitive}}$.

where $\langle V \rangle$ is the average voltage. In the limit as $p \to \infty$, the norm simply becomes the difference between the action potential's peak voltage and the mean voltage, whereas a finite $p$ ensures that the norm is differentiable. In parameter space, we will focus our attention to the manifold of action potentials with constant $L^p$ norm with $2 \ll p < \infty$, which entails that the optimal action potential will have a finite, though possibly narrow width. To be close to the supremum norm, yet still have a norm that is well-behaved under differentiation, we decided to use $p = 16$.

## 5 Poincaré-Lindstedt perturbation of periodic dynamical orbits

Standard (secular) perturbation theory diverges for periodic orbits, so we apply the Poincar-Lindstedt technique of expanding both in the period and the dynamics of the asymptotic orbit and then derive a set of adjoint sensitivity equations for the differential-algebraic system. Solving once for the adjoint functions, we can easily compute the parameter gradient of any functional on the orbit, even for thousands of parameters.

We start with a set of ordinary differential equations $\dot{\mathbf{x}} = \mathbf{F}(\mathbf{x}; \mathbf{p})$ for the neuron's dynamics, an asymptotically periodic orbit $\mathbf{x}^\gamma(t)$ that describes the action potential, and a functional $G(\mathbf{x}; \mathbf{p})$ on the orbit, representing the energy consumption, for instance. The functional can be written as an integral

$$G(\mathbf{x}^\gamma; \mathbf{p}) = \int_0^{\omega(\mathbf{p})^{-1}} g(\mathbf{x}^\gamma(t); \mathbf{p}) \, dt,$$

over some source term $g(\mathbf{x}^\gamma(t); \mathbf{p})$. Assume that locally perturbing a parameter $p \in \mathbf{p}$ induces a smooth change in the stable limit cycle, preserving its existence. Generally, a perturbation changes not only the limit cycle's path in state space, but also the average speed with which this orbit is traversed; as a consequence, the value of the functional depends on this change in speed, to lowest order. For simplicity, consider a single, scalar parameter $p$. $G(\mathbf{x}^\gamma; p)$ is the solution to

$$\omega(p) \partial_\tau \left[ G(\mathbf{x}^\gamma; p) \right] = g(\mathbf{x}^\gamma; p),$$

where we have normalised time via $\tau = \omega(p)t$. Denoting partial derivatives by subscripts, we expand $p \mapsto p + \epsilon$ to get the $\mathcal{O}\left(\epsilon^1\right)$ equation

$$d_\tau \left[ G_p(\mathbf{x}^\gamma; p) \right] + \omega_p g(\mathbf{x}^\gamma; p) = g_{\mathbf{x}}(\mathbf{x}^\gamma; p) \mathbf{x}_p + g_p(\mathbf{x}^\gamma; p)$$

in a procedure known as the Poincaré-Lindstedt method. Hence,

$$\frac{dG}{dp} = \int_0^{\omega^{-1}} \left( g_p + g_{\mathbf{x}} \mathbf{x}_p - \omega_p g \right) dt,$$

where, once again by the Poincaré-Lindstedt method, $\mathbf{x}_p$ is the solution to

$$\dot{\mathbf{x}}_p = \mathbf{F}_{\mathbf{x}}(\mathbf{x}^\gamma) \mathbf{x}_p + \mathbf{F}_p(\mathbf{x}^\gamma) - \omega_p \mathbf{F}(\mathbf{x}^\gamma).$$

Following the approach described by Cao, Li, Petzold, and Serban (2003), introduce a Lagrange *vector* $\mathbf{A}^G(\mathbf{x})$ and consider the augmented objective function

$$I(\mathbf{x}^\gamma; p) = G(\mathbf{x}^\gamma; p) - \int_0^{\omega^{-1}} \mathbf{A}^G(\mathbf{x}^\gamma) . \left( \mathbf{F}(\mathbf{x}^\gamma) - \dot{\mathbf{x}}^\gamma \right) dt,$$

which is identical to $G(\mathbf{x}^\gamma; p)$ as $\mathbf{F}(\mathbf{x}) - \dot{\mathbf{x}} = 0$. Then

$$\frac{dI(\mathbf{x}^\gamma; p)}{dp} = \int_0^{\omega^{-1}} \left( g_p + g_{\mathbf{x}} \mathbf{x}_p - \omega_p g \right) dt - \int_0^{\omega^{-1}} \mathbf{A}^G . \left( \mathbf{F}_p + \mathbf{F}_{\mathbf{x}} \mathbf{x}_p - \omega_p \mathbf{F} - \dot{\mathbf{x}}_p \right) dt.$$

Integrating the $\mathbf{A}^G(\mathbf{x}) . \dot{\mathbf{x}}_p$ term by parts and using periodicity, we get

$$\frac{dI(\mathbf{x}^\gamma; p)}{dp} = \int_0^{\omega^{-1}} \left[ g_p - \omega_p g - \mathbf{A}^G . \left( \mathbf{F}_p - \omega_p \mathbf{F} \right) \right] dt - \int_0^{\omega^{-1}} \left[ -g_{\mathbf{x}} + \dot{\mathbf{A}}^G + \mathbf{A}^G . \mathbf{F} \right] \mathbf{x}_p \, dt.$$

| Parameter | minimum | maximum |
|---|---|---|
| peak permeability $\bar{P}_{\text{Na}}$ | 0.24 fm/s | 0.15 $\mu$m/s |
| peak permeability $\bar{P}_{\text{K}}$ | 6.6 fm/s | 11 $\mu$m/s |
| midpoint voltage $V_m \vee V_h$ | - 72 mV | 70 mV |
| slope $s_m \vee (-s_h)$ | 3.33 mV | 200 mV |
| time constant $\tau_{m,0} \vee \tau_{h,0}$ | 5 $\mu$s | 200 ms |
| gating exponent $r \vee s$ | 0.2 | 5.0 |

Table 1: Parameter limits.

We can let the second term vanish by making the vector $\mathbf{A}^G(\mathbf{x})$ obey

$$\dot{\mathbf{A}}^G(\mathbf{x}) = -\mathbf{F}_{\mathbf{x}}^T(\mathbf{x}; p)\,\mathbf{A}^G(\mathbf{x}) + g_{\mathbf{x}}(\mathbf{x}; p).$$

Label the homogeneous solution (obtained by setting $g_{\mathbf{x}}(\mathbf{x}^\gamma; p) = 0$) as $\mathbf{Z}(\mathbf{x})$. It is known that the term $\omega_p$ is given by $\omega_p = \omega \int_0^{\omega^{-1}} \mathbf{Z}(\mathbf{x}).\mathbf{F}_p(\mathbf{x})\,dt$, provided $\mathbf{Z}(\mathbf{x})$ is normalised to satisfy $\mathbf{Z}(\mathbf{x}).\mathbf{F}(\mathbf{x}) = 1$. We can add any multiple of the homogeneous solution $\mathbf{Z}(\mathbf{x})$ to the inhomogeneous solution, so we can always make

$$\int_0^{\omega^{-1}} \mathbf{A}^G(\mathbf{x}).\mathbf{F}(\mathbf{x})\,dt = G$$

by taking

$$\mathbf{A}^G(\mathbf{x}) \mapsto \mathbf{A}^G(\mathbf{x}) - \mathbf{Z}(\mathbf{x})\left(\int_0^{\omega^{-1}} \mathbf{A}^G(\mathbf{x}).\mathbf{F}(\mathbf{x})\,dt - \omega G\right). \tag{3}$$

This condition will make $\mathbf{A}^G(\mathbf{x})$ unique. Finally, with eq. (3) we get

$$\frac{dG(\mathbf{x}^\gamma; p)}{dp} = \frac{dI(\mathbf{x}^\gamma; p)}{dp} = \int_0^{\omega^{-1}} \left(g_p - \mathbf{A}^G.\mathbf{F}_p\right)dt.$$

The first term in the integral gives rise to the partial derivative $\partial G(\mathbf{x}^\gamma; p)/\partial p$. In many cases, this term is either zero, can be made zero, or at least made independent of the dynamical variables.

The parameters for the neuron models are listed in Table 1 together with their minimum and maximum allowed values.

For each parameter in the neuron model, an auxiliary parameter on the entire real line is introduced, and a mapping from the real line onto the finite range set by the biophysical limits is defined. Gradient descent on this auxiliary parameter space is performed by orthogonalizing the gradient $dQ_\alpha/dp$ to the gradient $dL/dp$ of the norm. To correct for drift off the constraint manifold of constant norm, illustrated in Fig. 3, steps of gradient ascent or descent on the $L^p$ norm are performed while keeping $Q_\alpha$ constant. The step size during gradient descent is adjusted to assure that $\Delta Q_\alpha < 0$ and that a periodic solution $\mathbf{x}^\gamma$ exists after adapting the parameters. The energy landscape is locally convex (Fig. 3).

## 6 Predicting the Hodgkin-Huxley model

We start with a single-compartment Goldman-Hodgkin-Katz model neuron containing voltage-gated Na$^+$ and leak conductances (Figure 1). A tonic synaptic input to the model evokes repetitive firing of action potentials. We seek those parameters that minimize the ionic load for an action potential of constant norm—in other words, spikes whose height relative to the average voltage is fairly constant, subject to a trade-off with the spike width. The ionic load is directly proportional to the work $W$ performed by the ion flux. All parameters governing the ion channels' voltage dependence and kinetics, including their time constants, mid-points, slopes, and peak values, are subject to change.

The simplest model capable of generating an action potential must have two dynamical variables and two time scales: one for the upstroke and another for the downstroke. If both Na$^+$ and K$^+$ currents

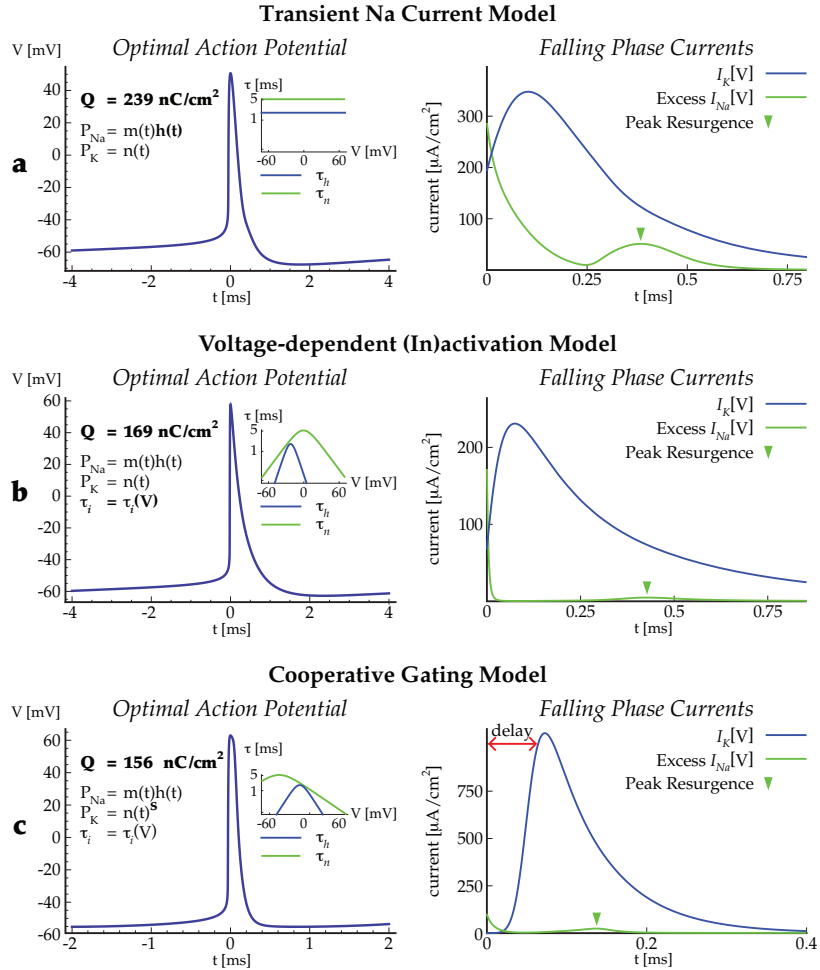

Figure 2: Optimal spike shapes and currents for neuron models with different biophysical features. During optimization, the spikes were constrained to have constant norm $\|V(t) - \langle V \rangle\|_{16} = 92$ mV, which controls the height of the spike. Insets in the left column display the voltage-dependence of the optimized time constants for sodium inactivation and potassium activation; sodium activation is modeled as occurring instantaneously. (a) Model with voltage-dependent inactivation of $Na^+$; time constants for the first order permeability kinetics are voltage-independent (inset). Inactivation turns off the $Na^+$ current on the downstroke, but not completely: as the $K^+$ current activates to repolarize the membrane, the inward $Na^+$ current reactivates and counteracts the $K^+$ current; the peak of the resurgent $Na^+$ current is marked by a triangle. (b) Model with voltage-dependent time constants for the first order kinetics of activation and inactivation. The voltage dependence minimizes the resurgence of the $Na^+$ current. (c) Power-law gating model with an inwardly rectifying potassium current replacing the leak current. The power law dependence introduces an effective delay in the onset of the $K^+$ current, which further minimizes the overlap of $Na^+$ and $K^+$ currents in time.

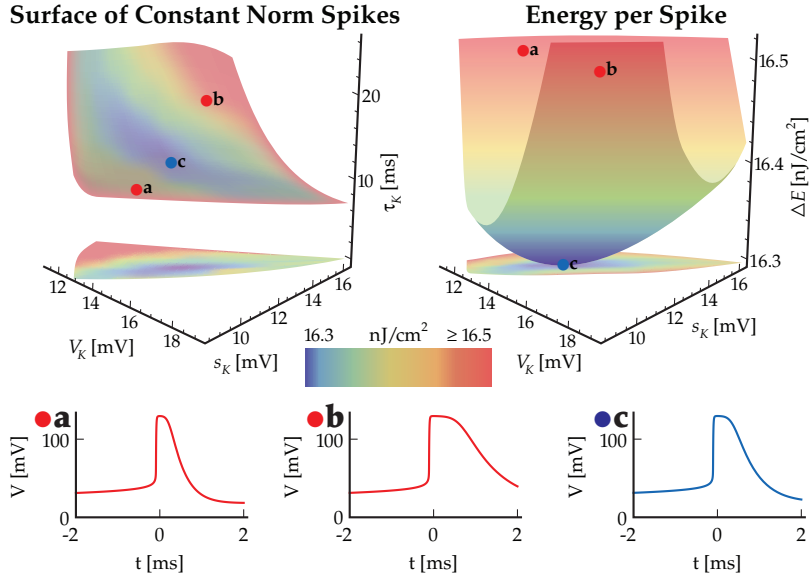

Figure 3: The energy required for an action potential three parameters governing potassium activation: the midpoint voltage $V_K$, the slope $s_K$, and the (maximum) time constant $\tau_K$. The energy is the minimum work required to restore the ionic concentration gradients, as given by Eq. (1). Note that the energy within the constrained manifold of constant norm spikes is locally convex.

are persistent, current flows in opposite directions at the same time, so that, even at the optimum, the ionic load is 1200 nC/cm². On the other hand, no voltage-gated K⁺ channels are even required for a spike, as long as Na⁺ channels activate on a fast time scale and inactivate on a slower time scale and the leak is powerful enough to repolarize the neuron. Even so, the load is still 520 nC/cm².

While spikes require dynamics on two time scales, suppressing the overlap between inward and outward currents calls for a third time scale. The resulting dynamics are higher-dimensional and reduce the load to to 239 nC/cm².

Making the activation and inactivation time constants voltage-dependent permits ion channels to latch to an open or closed state during the rising and falling phase of the spike, reducing the ionic load to 189 nC/cm² (Fig. 2) . The minimal Na⁺ and K⁺ currents are separated in time, yet dynamics that are linear in the activation variables cannot enforce a true delay between the offset of the Na⁺ current and the onset of the K⁺ current. If current flow depends on multiple gates that need to be activated simultaneously, optimization can use the nonlinearity of multiplication to introduce a delay in the rise of the K⁺ current that abolishes the overlap, and the ionic load drops to 156 nC/cm².

Any number of kinetic schemes for the nonlinear permeabilities $P_\alpha$ can give rise to the same spike waveform $V(t)$, including the simplest two-dimensional one. Yet only the full Hodgkin-Huxley (HH) model, with its voltage-dependent kinetics that prevent the premature resurgence of inward current and cooperative gating that delays the onset of the outward current, minimizes the energetic cost. More complex models, in which voltage-dependent ion channels make transitions between multiple closed, inactivated, and open states, instantiate the energy-conserving features of the HH system at the molecular level. Furthermore, features that are eliminated during optimization, such as a voltage-dependent inactivation of the outward potassium current, are also not part of the delayed rectifier potassium current in the Hodgkin-Huxley framework.

# References

[1] Paul De Weer, David C. Gadsby, and R. F. Rakowski. Voltage dependence of the na-k pump. *Ann. Rev. Physiol.*, 50:225–241, 1988.

[2] B. Frankenhaeuser and A. F. Huxley. The action potential in the myelinated nerve fibre of *xenopus laevis* as computed on the basis of voltage clamp data. *J. Physiol.*, 171:302–315, 1964.

[3] Samuel S.-H. Wang, Jennifer R. Shultz, Mark J. Burish, Kimberly H. Harrison, Patrick R. Hof, Lex C. Towns, Matthew W. Wagers, and Krysta D. Wyatt. Functional trade-offs in white matter axonal scaling. *J. Neurosci.*, 28(15):4047–4056, 2008.

[4] Henrik Alle, Arnd Roth, and Jörg R. P. Geiger. Energy-efficient action potentials in hippocampal mossy fibers. *Science*, 325(5946):1405–1408, 2009.

[5] D. E. Goldman. Potential, impedance and rectification in membranes. *J. Gen. Physiol.*, 27:37–60, 1943.

[6] A. L. Hodgkin and B. Katz. The effect of sodium ions on the electrical activity of the giant axon of the squid. *J. Physiol.*, 108:37–77, 1949.

[7] Brett C. Carter and Bruce P. Bean. Sodium entry during action potentials of mammalian neurons: Incomplete inactivation and reduced metabolic efficiency in fast-spiking neurons. *Neuron*, 64(6):898–909, 2009.

[8] Luc J. Gentet, Greg J. Stuart, and John D. Clements. Direct measurement of specific membrane capacitance in neurons. *Biophys. J.*, 79:314–320, 2000.

[9] Alain Destexhe, Michael Rudolph, and Denis Paré. The high-conductance state of neocortical neurons *in vivo*. *Nature Neurosci. Rev.*, 4:739–751, 2003.

[10] Bilal Haider and David A. McCormick. Rapid neocortical dynamics: Cellular and network mechanisms. *Neuron*, 62:171–189, 2009.

